# Is Learning The $n$-th Thing Any Easier Than Learning The First?

**Sebastian Thrun**[1]

Computer Science Department
Carnegie Mellon University
Pittsburgh, PA 15213-3891
World Wide Web: http://www.cs.cmu.edu/~thrun

## Abstract

This paper investigates learning in a lifelong context. Lifelong learning addresses situations in which a learner faces a whole stream of learning tasks. Such scenarios provide the opportunity to transfer knowledge across multiple learning tasks, in order to generalize more accurately from less training data. In this paper, several different approaches to lifelong learning are described, and applied in an object recognition domain. It is shown that across the board, lifelong learning approaches generalize consistently more accurately from less training data, by their ability to transfer knowledge across learning tasks.

## 1 Introduction

Supervised learning is concerned with approximating an unknown function based on examples. Virtually all current approaches to supervised learning assume that one is given a set of input-output examples, denoted by $X$, which characterize an unknown function, denoted by $f$. The target function $f$ is drawn from a class of functions, $F$, and the learner is given a space of hypotheses, denoted by $H$, and an order (preference/prior) with which it considers them during learning. For example, $H$ might be the space of functions represented by an artificial neural network with different weight vectors.

While this formulation establishes a rigid framework for research in machine learning, it dismisses important aspects that are essential for human learning. Psychological studies have shown that humans often employ more than just the training data for generalization. They are often able to generalize correctly even from a single training example [2, 10]. One of the key aspects of the learning problem faced by humans, which differs from the vast majority of problems studied in the field of neural network learning, is the fact that humans encounter a whole stream of learning problems over their entire lifetime. When faced with a new thing to learn, humans can usually exploit an enormous amount of training data and

experiences that stem from other, related learning tasks. For example, when learning to drive a car, years of learning experience with basic motor skills, typical traffic patterns, logical reasoning, language and much more precede and influence this learning task. The transfer of knowledge across learning tasks seems to play an essential role for generalizing accurately, particularly when training data is scarce.

A framework for the study of the transfer of knowledge is the *lifelong learning framework*. In this framework, it is assumed that a learner faces a whole collection of learning problems over its entire lifetime. Such a scenario opens the opportunity for synergy. When facing its *n*-th learning task, a learner can re-use knowledge gathered in its previous $n - 1$ learning tasks to boost the generalization accuracy.

In this paper we will be interested in the most simple version of the lifelong learning problem, in which the learner faces a family of *concept learning tasks*. More specifically, the functions to be learned over the lifetime of the learner, denoted by $f_1, f_2, f_3, \ldots \in F$, are all of the type $f : I \longrightarrow \{0, 1\}$ and sampled from $F$. Each function $f \in \{f_1, f_2, f_3, \ldots\}$ is an indicator function that defines a particular concept: a pattern $x \in I$ is member of this concept if and only if $f(x) = 1$. When learning the *n*-th indicator function, $f_n$, the training set $X$ contains examples of the type $\langle x, f_n(x) \rangle$ (which may be distorted by noise). In addition to the training set, the learner is also given $n - 1$ sets of examples of other concept functions, denoted by $X_k$ ($k = 1, \ldots, n - 1$). Each $X_k$ contains training examples that characterize $f_k$. Since this additional data is desired to support learning $f_n$, $X_k$ is called a *support set* for the training set $X$.

An example of the above is the recognition of faces [5, 7]. When learning to recognize the *n*-th person, say $f_{\text{Bob}}$, the learner is given a set of positive and negative example of face images of this person. In lifelong learning, it may also exploit training information stemming from other persons, such as $f \in \{f_{\text{Rich}}, f_{\text{Mike}}, f_{\text{Dave}}, \ldots\}$. The support sets usually cannot be used directly as training patterns when learning a new concept, since they describe different concepts (hence have different class labels). However, certain features (like the shape of the eyes) are more important than others (like the facial expression, or the location of the face within the image). Once the invariances of the domain are learned, they can be transferred to new learning tasks (new people) and hence improve generalization.

To illustrate the potential importance of related learning tasks in lifelong learning, this paper does not present just one particular approach to the transfer of knowledge. Instead, it describes several, all of which extend conventional memory-based or neural network algorithms. These approaches are compared with more traditional learning algorithms, *i.e.*, those that do not transfer knowledge. The goal of this research is to demonstrate that, independent of a particular learning approach, more complex functions can be learned from less training data if learning is embedded into a lifelong context.

## 2   Memory-Based Learning Approaches

Memory-based algorithms memorize all training examples explicitly and interpolate them at query-time. We will first sketch two simple, well-known approaches to memory-based learning, then propose extensions that take the support sets into account.

### 2.1   Nearest Neighbor and Shepard's Method

Probably the most widely used memory-based learning algorithm is *K-nearest neighbor (KNN)* [15]. Suppose $x$ is a query pattern, for which we would like to know the output $y$. KNN searches the set of training examples $X$ for those $K$ examples $\langle x_i, y_i \rangle \in X$ whose input patterns $x_i$ are nearest to $x$ (according to some distance metric, *e.g.*, the Euclidian distance). It then returns the mean output value $\frac{1}{K} \sum y_i$ of these nearest neighbors.

Another commonly used method, which is due to Shepard [13], averages the output values

of *all* training examples but weights each example according to the inverse distance to the query point $x$.

$$s(x) \quad := \quad \left( \sum_{\langle x_i, y_i \rangle \in X} \frac{y_i}{||x - x_i|| + \varepsilon} \right) \cdot \left( \sum_{\langle x_i, y_i \rangle \in X} \frac{1}{||x - x_i|| + \varepsilon} \right)^{-1} \quad (1)$$

Here $\varepsilon > 0$ is a small constant that prevents division by zero. Plain memory-based learning uses exclusively the training set $X$ for learning. There is no obvious way to incorporate the support sets, since they carry the wrong class labels.

## 2.2 Learning A New Representation

The first modification of memory-based learning proposed in this paper employs the support sets to learn a *new representation* of the data. More specifically, the support sets are employed to learn a function, denoted by $g : I \longrightarrow I'$, which maps input patterns in $I$ to a new space, $I'$. This new space $I'$ forms the input space for a memory-based algorithm.

Obviously, the key property of a good data representations is that multiple examples of a single concept should have a similar representation, whereas the representation of an example and a counterexample of a concept should be more different. This property can directly be transformed into an energy function for $g$:

$$E := \sum_{k=1}^{n-1} \sum_{\langle x, y=1 \rangle \in X_k} \left( \sum_{\langle x', y'=1 \rangle \in X_k} ||g(x) - g(x')|| - \sum_{\langle x', y'=0 \rangle \in X_k} ||g(x) - g(x')|| \right) \quad (2)$$

Adjusting $g$ to minimize $E$ forces the distance between pairs of examples of the same concept to be small, and the distance between an example and a counterexample of a concept to be large. In our implementation, $g$ is realized by a neural network and trained using the Back-Propagation algorithm [12].

Notice that the new representation, $g$, is obtained through the support sets. Assuming that the learned representation is appropriate for new learning tasks, standard memory-based learning can be applied using this new representation when learning the $n$-th concept.

## 2.3 Learning A Distance Function

An alternative way for exploiting support sets to improve memory-based learning is to learn a distance function [3, 9]. This approach learns a function $d : I \times I \longrightarrow [0, 1]$ which accepts two input patterns, say $x$ and $x'$, and outputs whether $x$ and $x'$ are members of the same concept, regardless what the concept is. Training examples for $d$ are

$$\langle (x, x'), 1 \rangle \quad \text{if } y = y' = 1$$
$$\langle (x, x'), 0 \rangle \quad \text{if } (y = 1 \wedge y' = 0) \text{ or } (y = 0 \wedge y' = 1) .$$

They are derived from pairs of examples $\langle x, y \rangle, \langle x', y' \rangle \in X_k$ taken from a single support set $X_k$ ($k = 1, \ldots, n - 1$). In our implementation, $d$ is an artificial neural network trained with Back-Propagation. Notice that the training examples for $d$ lack information concerning the concept for which they were originally derived. Hence, all support sets can be used to train $d$. After training, $d$ can be interpreted as the probability that two patterns $x, x' \in I$ are examples of the same concept.

Once trained, $d$ can be used as a *generalized distance function* for a memory-based approach. Suppose one is given a training set $X$ and a query point $x \in I$. Then, for each positive example $\langle x', y' = 1 \rangle \in X$, $d(x, x')$ can be interpreted as the probability that $x$ is a member of the target concept. Votes from multiple positive examples $\langle x_1, 1 \rangle, \langle x_2, 1 \rangle, \ldots \in X$ are combined using Bayes' rule, yielding

$$Prob(f_n(x) = 1) \quad := \quad 1 - \left( 1 + \prod_{\langle x', y'=1 \rangle \in X_k} \frac{d(x, x')}{1 - d(x, x')} \right)^{-1} \quad (3)$$

Notice that $d$ is not a distance metric. It generalizes the notion of a distance metric, because the triangle inequality needs not hold, and because an example of the target concept $x'$ can provide evidence that $x$ is *not* a member of that concept (if $d(x, x') < 0.5$).

# 3 Neural Network Approaches

To make our comparison more complete, we will now briefly describe approaches that rely exclusively on artificial neural networks for learning $f_n$.

## 3.1 Back-Propagation

Standard Back-Propagation can be used to learn the indicator function $f_n$, using $X$ as training set. This approach does not employ the support sets, hence is unable to transfer knowledge across learning tasks.

## 3.2 Learning With Hints

*Learning with hints* [1, 4, 6, 16] constructs a neural network with $n$ output units, one for each function $f_k$ ($k = 1, 2, \ldots, n$). This network is then trained to simultaneously minimize the error on both the support sets $\{X_k\}$ and the training set $X$. By doing so, the internal representation of this network is not only determined by $X$ but also shaped through the support sets $\{X_k\}$. If similar internal representations are required for all functions $f_k$ ($k = 1, 2, \ldots, n$), the support sets provide additional training examples for the internal representation.

## 3.3 Explanation-Based Neural Network Learning

The last method described here uses the *explanation-based neural network learning algorithm* (EBNN), which was originally proposed in the context of reinforcement learning [8, 17]. EBNN trains an artificial neural network, denoted by $h : I \longrightarrow [0, 1]$, just like Back-Propagation. However, in addition to the target values given by the training set $X$, EBNN estimates the *slopes* (tangents) of the target function $f_n$ for each example in $X$. More specifically, training examples in EBNN are of the sort $\langle x, f_n(x), \nabla_x f_n(x)\rangle$, which are fit using the Tangent-Prop algorithm [14]. The input $x$ and target value $f_n(x)$ are taken from the training set $X$. The third term, the slope $\nabla_x f_n(x)$, is estimated using the learned distance function $d$ described above. Suppose $\langle x', y' = 1\rangle \in X$ is a (positive) training example. Then, the function $d_{x'} : I \longrightarrow [0, 1]$ with $d_{x'}(z) := d(z, x')$ maps a single input pattern to $[0, 1]$, and is an approximation to $f_n$. Since $d(z, x')$ is represented by a neural network and neural networks are differentiable, the gradient $\partial d_{x'}(z)/\partial z$ is an estimate of the slope of $f_n$ at $z$. Setting $z := x$ yields the desired estimate of $\nabla_x f_n(x)$. As stated above, both the target value $f_n(x)$ and the slope vector $\nabla_x f_n(x)$ are fit using the Tangent-Prop algorithm for each training example $x \in X$.

The slope $\nabla_x f_n$ provides additional information about the target function $f_n$. Since $d$ is learned using the support sets, EBNN approach transfers knowledge from the support sets to the new learning task. EBNN relies on the assumption that $d$ is accurate enough to yield helpful sensitivity information. However, since EBNN fits both training patterns (values) and slopes, misleading slopes can be overridden by training examples. See [17] for a more detailed description of EBNN and further references.

# 4 Experimental Results

All approaches were tested using a database of color camera images of different objects (see Fig. 3.3). Each of the object in the database has a distinct color or size. The $n$-th

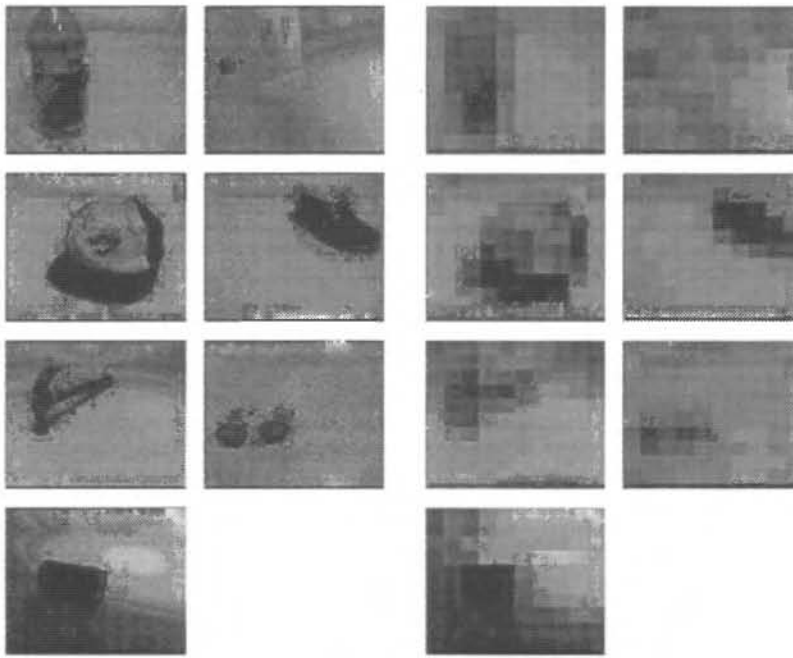

Figure 1: The support sets were compiled out of a hundred images of a *bottle*, a *hat*, a *hammer*, a *coke can*, and a *book*. The $n$-th learning tasks involves distinguishing the *shoe* from the *sunglasses*. Images were subsampled to a $100 \times 100$ pixel matrix (each pixel has a color, saturation, and a brightness value), shown on the right side.

learning task was the recognition of one of these objects, namely the *shoe*. The previous $n - 1$ learning tasks correspond to the recognition of five other objects, namely the *bottle*, the *hat*, the *hammer*, the *coke can*, and the *book*. To ensure that the latter images could not be used simply as additional training data for $f_n$, the only counterexamples of the *shoe* was the seventh object, the *sunglasses*. Hence, the training set for $f_n$ contained images of the *shoe* and the *sunglasses*, and the support sets contained images of the other five objects. The object recognition domain is a good testbed for the transfer of knowledge in lifelong learning. This is because finding a good approximation to $f_n$ involves recognizing the target object invariant of rotation, translation, scaling in size, change of lighting and so on. Since these invariances are common to all object recognition tasks, images showing other objects can provide additional information and boost the generalization accuracy.

Transfer of knowledge is most important when training data is scarce. Hence, in an initial experiment we tested all methods using a single image of the *shoe* and the *sunglasses* only. Those methods that are able to transfer knowledge were also provided 100 images of each of the other five objects. The results are intriguing. The generalization accuracies

| KNN | Shepard | **repr. $g$+Shep.** | **distance $d$** |
|---|---|---|---|
| 60.4% | 60.4% | **74.4%** | **75.2%** |
| ±8.3% | ±8.3% | **±18.5%** | **±18.9%** |

| Back-Prop | **hints** | **EBNN** |
|---|---|---|
| 59.7% | **62.1%** | **74.8%** |
| ±9.0% | **±10.2%** | **±11.1%** |

illustrate that all approaches that transfer knowledge (printed in bold font) generalize significantly better than those that do not. With the exception of the hint learning technique, the approaches can be grouped into two categories: Those which generalize approximately 60% of the testing set correctly, and those which achieve approximately 75% generalization accuracy. The former group contains the standard supervised learning algorithms, and the latter contains the "new" algorithms proposed here, which are capable of transferring knowledge. The differences within each group are statistically not significant, while the differences between them are (at the 95% level). Notice that random guessing classifies 50% of the testing examples correctly.

These results suggest that the generalization accuracy merely depends on the particular choice of the learning algorithm (memory-based vs. neural networks). Instead, the main factor determining the generalization accuracy is the fact whether or not knowledge is transferred from past learning tasks.

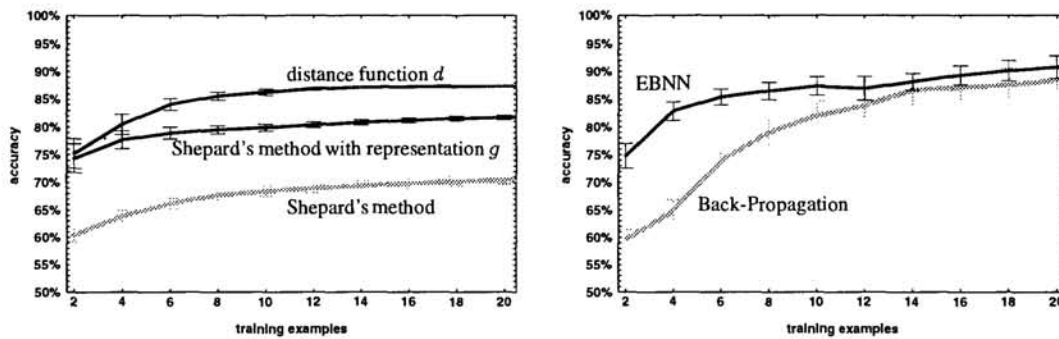

Figure 2: Generalization accuracy as a function of training examples, measured on an independent test set and averaged over 100 experiments. 95%-confidence bars are also displayed.

What happens as more training data arrives? Fig. 2 shows generalization curves with increasing numbers of training examples for some of these methods. As the number of training examples increases, prior knowledge becomes less important. After presenting 20 training examples, the results

| KNN | Shepard | repr. $g$+Shep. | distance $d$ | | Back-Prop | hints | EBNN |
|---|---|---|---|---|---|---|---|
| 81.0% | 70.5% | **81.7%** | **87.3%** | | 88.4% | **n.avail.** | **90.8%** |
| ±3.4% | ±4.9% | **±2.7%** | **±0.9%** | | ±2.5% | | **±2.7%** |

illustrate that some of the standard methods (especially Back-Propagation) generalize about as accurately as those methods that exploit support sets. Here the differences in the underlying learning mechanisms becomes more dominant. However, when comparing lifelong learning methods with their corresponding standard approaches, the latter ones are still inferior: Back-Propagation (88.4%) is outperformed by EBNN (90.8%), and Shepard's method (70.5%) generalizes less accurately when the representation is learned (81.7%) or when the distance function is learned (87.3%). All these differences are significant at the 95% confidence level.

## 5 Discussion

The experimental results reported in this paper provide evidence that learning becomes easier when embedded in a lifelong learning context. By transferring knowledge across related learning tasks, a learner can become "more experienced" and generalize better. To test this conjecture in a more systematic way, a variety of learning approaches were evaluated and compared with methods that are unable to transfer knowledge. It is consistently found that lifelong learning algorithms generalize significantly more accurately, particularly when training data is scarce.

Notice that these results are well in tune with other results obtained by the author. One of the approaches here, EBNN, has extensively been studied in the context of robot perception [11], reinforcement learning for robot control, and chess [17]. In all these domains, it has consistently been found to generalize better from less training data by transferring knowledge from previous learning tasks. The results are also consistent with observations made about human learning [2, 10], namely that previously learned knowledge plays an important role in generalization, particularly when training data is scarce. [18] extends these techniques to situations where most support sets are not related.w

However, lifelong learning rests on the assumption that more than a single task is to be learned, and that learning tasks are appropriately related. Lifelong learning algorithms are particularly well-suited in domains where the costs of collecting training data is the dominating factor in learning, since these costs can be amortized over several learning tasks. Such domains include, for example, autonomous service robots which are to learn and improve over their entire lifetime. They include personal software assistants which have

to perform various tasks for various users. Pattern recognition, speech recognition, time series prediction, and database mining might be other, potential application domains for the techniques presented here.

## Footnotes

[1] also affiliated with: Institut für Informatik III, Universität Bonn, Römerstr. 164, Germany

# References

[1] Y. S. Abu-Mostafa. Learning from hints in neural networks. *Journal of Complexity*, 6:192–198, 1990.

[2] W.-K. Ahn and W. F. Brewer. Psychological studies of explanation-based learning. In G. DeJong, editor, *Investigating Explanation-Based Learning*. Kluwer Academic Publishers, Boston/Dordrecht/London, 1993.

[3] C. A. Atkeson. Using locally weighted regression for robot learning. In *Proceedings of the 1991 IEEE International Conference on Robotics and Automation*, pages 958–962, Sacramento, CA, April 1991.

[4] J. Baxter. Learning internal representations. In *Proceedings of the Conference on Computation Learning Theory*, 1995.

[5] D. Beymer and T. Poggio. Face recognition from one model view. In *Proceedings of the International Conference on Computer Vision*, 1995.

[6] R. Caruana. Multitask learning: A knowledge-based of source of inductive bias. In P. E. Utgoff, editor, *Proceedings of the Tenth International Conference on Machine Learning*, pages 41–48, San Mateo, CA, 1993. Morgan Kaufmann.

[7] M. Lando and S. Edelman. Generalizing from a single view in face recognition. Technical Report CS-TR 95-02, Department of Applied Mathematics and Computer Science, The Weizmann Institute of Science, Rehovot 76100, Israel, January 1995.

[8] T. M. Mitchell and S. Thrun. Explanation-based neural network learning for robot control. In S. J. Hanson, J. Cowan, and C. L. Giles, editors, *Advances in Neural Information Processing Systems 5*, pages 287–294, San Mateo, CA, 1993. Morgan Kaufmann.

[9] A. W. Moore, D. J. Hill, and M. P. Johnson. An Empirical Investigation of Brute Force to choose Features, Smoothers and Function Approximators. In S. Hanson, S. Judd, and T. Petsche, editors, *Computational Learning Theory and Natural Learning Systems, Volume 3*. MIT Press, 1992.

[10] Y. Moses, S. Ullman, and S. Edelman. Generalization across changes in illumination and viewing position in upright and inverted faces. Technical Report CS-TR 93-14, Department of Applied Mathematics and Computer Science, The Weizmann Institute of Science, Rehovot 76100, Israel, 1993.

[11] J. O'Sullivan, T. M. Mitchell, and S. Thrun. Explanation-based neural network learning from mobile robot perception. In K. Ikeuchi and M. Veloso, editors, *Symbolic Visual Learning*. Oxford University Press, 1995.

[12] D. E. Rumelhart, G. E. Hinton, and R. J. Williams. Learning internal representations by error propagation. In D. E. Rumelhart and J. L. McClelland, editors, *Parallel Distributed Processing. Vol. I + II*. MIT Press, 1986.

[13] D. Shepard. A two-dimensional interpolation function for irregularly spaced data. In *23rd National Conference ACM*, pages 517–523, 1968.

[14] P. Simard, B. Victorri, Y. LeCun, and J. Denker. Tangent prop – a formalism for specifying selected invariances in an adaptive network. In J. E. Moody, S. J. Hanson, and R. P. Lippmann, editors, *Advances in Neural Information Processing Systems 4*, pages 895–903, San Mateo, CA, 1992. Morgan Kaufmann.

[15] C. Stanfill and D. Waltz. Towards memory-based reasoning. *Communications of the ACM*, 29(12):1213–1228, December 1986.

[16] S. C. Suddarth and A. Holden. Symbolic neural systems and the use of hints for developing complex systems. *International Journal of Machine Studies*, 35, 1991.

[17] S. Thrun. *Explanation-Based Neural Network Learning: A Lifelong Learning Approach*. Kluwer Academic Publishers, Boston, MA, 1996. to appear.

[18] S. Thrun and J. O'Sullivan. Clustering learning tasks and the selective cross-task transfer of knowledge. Technical Report CMU-CS-95-209, Carnegie Mellon University, School of Computer Science, Pittsburgh, PA 15213, November 1995.
